# Stochastic Mixed-Signal VLSI Architecture for High-Dimensional Kernel Machines

**Roman Genov**   and   **Gert Cauwenberghs**
Department of Electrical and Computer Engineering
Johns Hopkins University, Baltimore, MD 21218
{*roman,gert*}*@jhu.edu*

## Abstract

A mixed-signal paradigm is presented for high-resolution parallel inner-product computation in very high dimensions, suitable for efficient implementation of kernels in image processing. At the core of the externally digital architecture is a high-density, low-power analog array performing binary-binary partial matrix-vector multiplication. Full digital resolution is maintained even with low-resolution analog-to-digital conversion, owing to random statistics in the analog summation of binary products. A random modulation scheme produces near-Bernoulli statistics even for highly correlated inputs. The approach is validated with real image data, and with experimental results from a CID/DRAM analog array prototype in 0.5 $\mu$m CMOS.

## 1   Introduction

Analog computational arrays [1, 2, 3, 4] for neural information processing offer very large integration density and throughput as needed for real-time tasks in computer vision and pattern recognition [5]. Despite the success of adaptive algorithms and architectures in reducing the effect of analog component mismatch and noise on system performance [6, 7], the precision and repeatability of analog VLSI computation under process and environmental variations is inadequate for some applications. Digital implementation [10] offers absolute precision limited only by wordlength, but at the cost of significantly larger silicon area and power dissipation compared with dedicated, fine-grain parallel analog implementation, *e.g.,* [2, 4].

The purpose of this paper is twofold: to present an internally analog, externally digital architecture for dedicated VLSI kernel-based array processing that outperforms purely digital approaches with a factor 100-10,000 in throughput, density and energy efficiency; and to provide a scheme for digital resolution enhancement that exploits Bernoulli random statistics of binary vectors. Largest gains in system precision are obtained for high input dimensions. The framework allows to operate at full digital resolution with relatively imprecise analog hardware, and with minimal cost in implementation complexity to randomize the input data.

The computational core of inner-product based kernel operations in image processing and

pattern recognition is that of vector-matrix multiplication (VMM) in high dimensions:

$$Y_m = \sum_{n=0}^{N-1} W_{mn} X_n \tag{1}$$

with $N$-dimensional input vector $X_n$, $M$-dimensional output vector $Y_m$, and $N \times M$ matrix elements $W_{mn}$. In artificial neural networks, the matrix elements $W_{mn}$ correspond to weights, or synapses, between neurons. The elements also represent templates $X_n{}^m = W_{mn}$ in a vector quantizer [8], or support vectors in a support vector machine [9]. In what follows we concentrate on VMM computation which dominates inner-product based[1] kernel computations for high vector dimensions.

## 2 The *Kerneltron*: A Massively Parallel VLSI Computational Array

### 2.1 Internally Analog, Externally Digital Computation

The approach combines the computational efficiency of analog array processing with the precision of digital processing and the convenience of a programmable and reconfigurable digital interface.

The digital representation is embedded in the analog array architecture, with inputs presented in bit-serial fashion, and matrix elements stored locally in bit-parallel form:

$$W_{mn} = \sum_{i=0}^{I-1} 2^{-i-1} w_{mn}^{(i)} \tag{2}$$

$$X_n = \sum_{j=0}^{J-1} 2^{-j-1} x_n^{(j)} \tag{3}$$

decomposing (1) into:

$$Y_m = \sum_{n=0}^{N-1} W_{mn} X_n = \sum_{i=0}^{I-1} \sum_{j=0}^{J-1} 2^{-i-j-2} Y_m^{(i,j)} \tag{4}$$

with binary-binary VMM partials:

$$Y_m^{(i,j)} = \sum_{n=0}^{N-1} w_{mn}^{(i)} x_n^{(j)} \ . \tag{5}$$

The key is to compute and accumulate the binary-binary partial products (5) using an analog VMM array, and to combine the quantized results in the digital domain according to (4). Digital-to-analog conversion at the input interface is inherent in the bit-serial implementation, and row-parallel analog-to-digital converters (ADCs) are used at the output interface to quantize $Y_m{}^{(i,j)}$. A 512 $\times$ 128 array prototype using CID/DRAM cells is shown in Figure 1 (a).

### 2.2 CID/DRAM Cell and Array

The unit cell in the analog array combines a CID computational element [12, 13] with a DRAM storage element. The cell stores one bit of a matrix element $w_{mn}{}^{(i)}$, performs a one-quadrant binary-binary multiplication of $w_{mn}{}^{(i)}$ and $x_n{}^{(j)}$ in (5), and accumulates

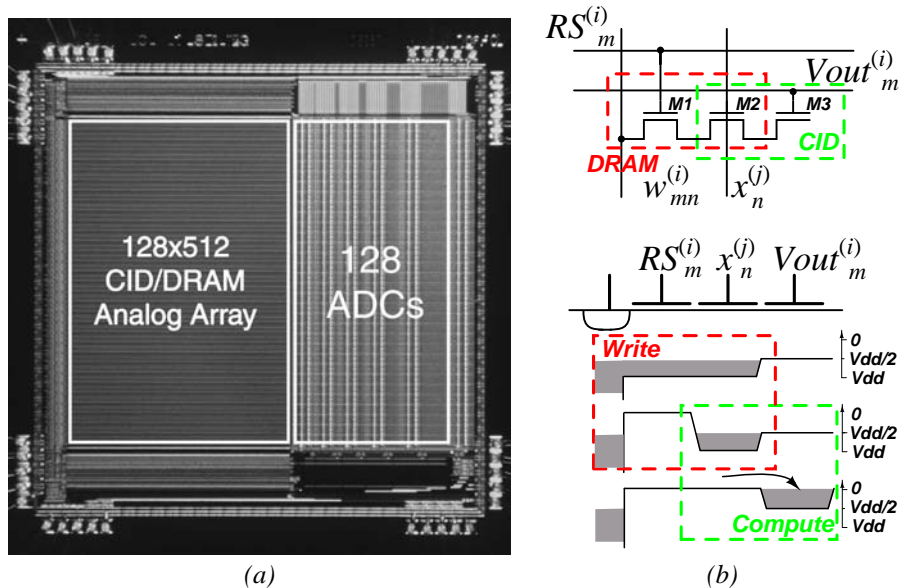

<div align="center">(a)          (b)</div>

Figure 1: *(a)* Micrograph of the *Kerneltron* prototype, containing an array of $512 \times 128$ CID/DRAM cells, and a row-parallel bank of 128 flash ADCs. Die size is $3 \text{ mm} \times 3 \text{ mm}$ in 0.5 $\mu$m CMOS technology. *(b)* CID computational cell with integrated DRAM storage. Circuit diagram, and charge transfer diagram for active write and compute operations.

the result across cells with common $m$ and $i$ indices. The circuit diagram and operation of the cell are given in Figure 1 (b). An array of cells thus performs (unsigned) binary multiplication (5) of matrix $w_{mn}^{(i)}$ and vector $x_n^{(j)}$ yielding $Y_m^{(i,j)}$, for values of $i$ in parallel across the array, and values of $j$ in sequence over time.

The cell contains three MOS transistors connected in series as depicted in Figure 1 (b). Transistors M1 and M2 comprise a dynamic random-access memory (DRAM) cell, with switch M1 controlled by *Row Select* signal $RS_m^{(i)}$. When activated, the binary quantity $w_{mn}^{(i)}$ is written in the form of charge (either $\triangle Q$ or 0) stored under the gate of M2. Transistors M2 and M3 in turn comprise a charge injection device (CID), which by virtue of charge conservation moves electric charge between two potential wells in a non-destructive manner [12, 13, 14].

The charge left under the gate of M2 can only be redistributed between the two CID transistors, M2 and M3. An active charge transfer from M2 to M3 can only occur if there is non-zero charge stored, and if the potential on the gate of M2 drops below that of M3 [12]. This condition implies a logical AND, *i.e.,* unsigned binary multiplication, of $w_{mn}^{(i)}$ and $x_n^{(j)}$. The multiply-and-accumulate operation is then completed by capacitively sensing the amount of charge transferred onto the electrode of M3, the output summing node. To this end, the voltage on the output line, left floating after being pre-charged to $Vdd/2$, is observed. When the charge transfer is active, the cell contributes a change in voltage $\triangle V_{out} = \triangle Q/C_{M3}$ where $C_{M3}$ is the total capacitance on the output line across cells. The total response is thus proportional to the number of actively transferring cells. After deactivating the input $x_n^{(j)}$, the transferred charge returns to the storage node M2. The CID computation is non-destructive and intrinsically reversible [12], and DRAM refresh is only required to counteract junction and subthreshold leakage.

The bottom diagram in Figure 1 (b) depicts the charge transfer timing diagram for write

and compute operations in the case when both $w_{mn}{}^{(i)}$ and $x_n{}^{(j)}$ are of logic level 1.

## 2.3 System-Level Performance

Measurements on the $512 \times 128$-element analog array and other fabricated prototypes show a dynamic range of 43 dB, and a computational cycle of 10 $\mu$s with power consumption of 50 nW per cell. The size of the CID/DRAM cell is $8\lambda \times 45\lambda$ with $\lambda = 0.3\mu m$.

The overall system resolution is limited by the precision in the quantization of the outputs from the analog array. Through digital postprocessing, two bits are gained over the resolution of the ADCs used [15], for a total system resolution of 8 bits. Larger resolutions can be obtained by accounting for the statistics of binary terms in the addition, the subject of the next section.

# 3 Resolution Enhancement Through Stochastic Encoding

Since the analog inner product (5) is discrete, zero error can be achieved (as if computed digitally) by matching the quantization levels of the ADC with each of the $N + 1$ discrete levels in the inner product. Perfect reconstruction of $Y_m{}^{(i,j)}$ from the quantized output, for an overall resolution of $I + J + \log_2(N+1)$ bits, assumes the combined effect of noise and nonlinearity in the analog array and the ADC is within one LSB (least significant bit). For large arrays, this places stringent requirements on analog precision and ADC resolution, $L \geq \log_2(N+1)$.

The implicit assumption is that all quantization levels are (equally) needed. A straightforward study of the statistics of the inner product, below, reveals that this is poor use of available resources.

## 3.1 Bernoulli Statistics

In what follows we assume signed, rather than unsigned, binary values for inputs and weights, $x_n{}^{(j)} = \pm1$ and $w_{mn}{}^{(i)} = \pm1$. This translates to exclusive-OR (XOR), rather than AND, multiplication on the analog array, an operation that can be easily accomplished with the CID/DRAM architecture by differentially coding input and stored bits using twice the number of columns and unit cells.

For input bits $x_n{}^{(j)}$ that are Bernoulli distributed (*i.e.,* fair coin flips), the (XOR) product terms $w_{mn}{}^{(i)} x_n{}^{(j)}$ in (5) are Bernoulli distributed, regardless of $w_{mn}{}^{(i)}$. Their sum $Y_m{}^{(i,j)}$ thus follows a binomial distribution

$$\Pr(Y_m^{(i,j)} = 2k - N) = \binom{N}{k} p^k (1-p)^{N-k} \tag{6}$$

with $p = 0.5$, $k = 0, ..., N$, which in the Central Limit $N \to \infty$ approaches a normal distribution with zero mean and variance $N$. In other words, for random inputs in high dimensions $N$ the active range (or standard deviation) of the inner-product is $N^{1/2}$, a factor $N^{1/2}$ smaller than the full range $N$.

In principle, this allows to relax the effective resolution of the ADC. However, any reduction in conversion range will result in a small but non-zero probability of overflow. In practice, the risk of overflow can be reduced to negligible levels with a few additional bits in the ADC conversion range. An alternative strategy is to use a variable resolution ADC which expands the conversion range on rare occurences of overflow.[2]

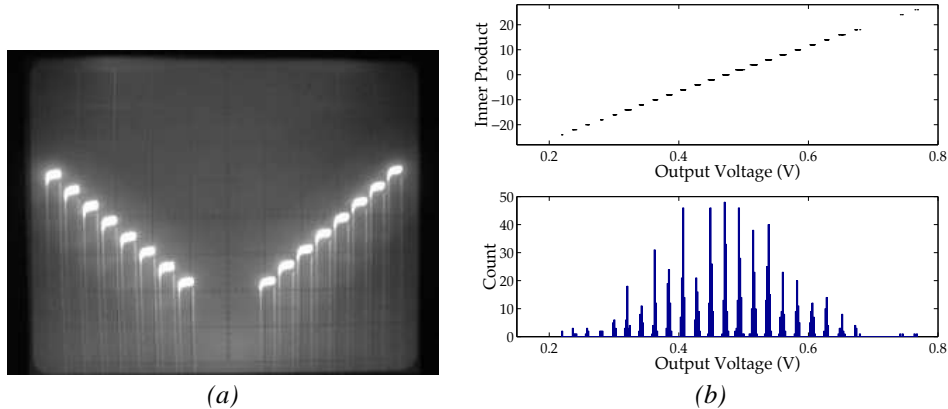

Figure 2: Experimental results from CID/DRAM analog array. *(a)* Output voltage on the sense line computing exclusive-or inner product of 64-dimensional stored and presented binary vectors. A variable number of active bits is summed at different locations in the array by shifting the presented bits. *(b) Top:* Measured output and actual inner product for 1,024 samples of Bernoulli distributed pairs of stored and presented vectors. *Bottom:* Histogram of measured array outputs.

## 3.2   Experimental Results

While the reduced range of the analog inner product supports lower ADC resolution in terms of number of quantization levels, it requires low levels of mismatch and noise so that the discrete levels can be individually resolved, near the center of the distribution. To verify this, we conducted the following experiment.

Figure 2 shows the measured outputs on one row of 128 CID/DRAM cells, configured differentially to compute signed binary (exclusive-OR) inner products of stored and presented binary vectors in 64 dimensions. The scope trace in Figure 2 (a) is obtained by storing all $+1$ bits, and shifting a sequence of input bits that differ with the stored bits by $32 \pm 4$ bits. The left and right segment of the scope trace correspond to different selections of active bit locations along the array that are maximally disjoint, to indicate a worst-case mismatch scenario. The measured and actual inner products in Figure 2 (b) are obtained by storing and presenting 1,024 pairs of random binary vectors. The histogram shows a clearly resolved, discrete binomial distribution for the observed analog voltage.

For very large arrays, mismatch and noise may pose a problem in the present implementation with floating sense line. A sense amplifier with virtual ground on the sense line and feedback capacitor optimized to the $N^{1/2}$ range would provide a simple solution.

## 3.3   Real Image Data

Although most randomly selected patterns do not correlate with any chosen template, patterns from the real world tend to correlate, and certainly those that are of interest to kernel computation [3]. The key is stochastic encoding of the inputs, as to randomize the bits presented to the analog array.

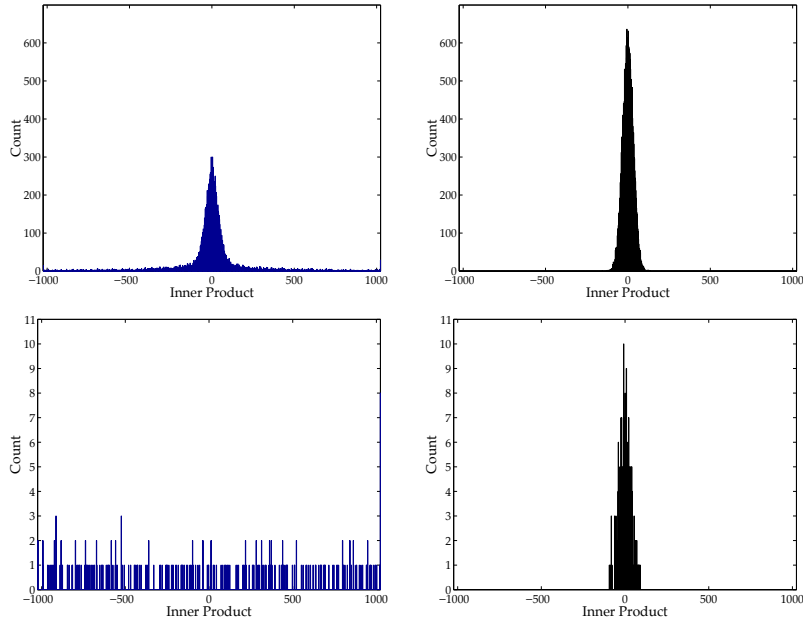

Figure 3: Histograms of partial binary inner products $Y^{(i,j)}$ for 256 pairs of randomly selected $32 \times 32$ pixel segments of *Lena*. *Left*: with unmodulated 8-bit image data for both vectors. *Right:* with 12-bit modulated stochastic encoding of one of the two vectors. *Top*: all bit planes $i$ and $j$. *Bottom*: most significant bit (MSB) plane, $i = j = 0$.

Randomizing an informative input while retaining the information is a futile goal, and we are content with a solution that approaches the ideal performance within observable bounds, and with reasonable cost in implementation. Given that "ideal" randomized inputs relax the ADC resolution by $log_2 N/2$ bits, they necessarily reduce the wordlenght of the output by the same. To account for the lost bits in the range of the output, it is necessary to increase the range of the "ideal" randomized input by the same number of bits.

One possible stochastic encoding scheme that restores the range is $N^{1/2}$-fold oversampling of the input through (digital) delta-sigma modulation. This is a workable solution; however we propose one that is simpler and less costly to implement. For each $I$-bit input component $X_n$, pick a random integer $U_n$ in the range $\pm(N^{1/2} - 1)$, and subtract it to produce a modulated input $\tilde{X}_n = X_n - U_n$ with $log_2 N/2$ additional bits. It can be shown that for worst-case deterministic inputs $X_n$ the mean of the inner product for $\tilde{X}_n$ is off at most by $\pm N^{1/2}$ from the origin. The desired inner products for $X_n$ are retrieved by digitally adding the inner products obtained for $\tilde{X}_n$ and $U_n$. The random offset $U_n$ can be chosen once, so its inner product with the templates can be pre-computed upon initializing or programming the array. The implementation cost is thus limited to component-wise subtraction of $X_n$ and $U_n$, achieved using one full adder cell, one bit register, and ROM storage of the $u_n^{(i)}$ bits for every column of the array.

Figure 3 provides a proof of principle, using image data selected at random from *Lena*. 12-bit stochastic encoding of the 8-bit image, by subtracting a random variable in a range 15 times larger than the image, produces the desired binomial distribution for the partial bit inner products, even for the most significant bit (MSB) which is most highly correlated.

# 4 Conclusions

We presented an externally digital, internally analog VLSI array architecture suitable for real-time kernel-based neural computation and machine learning in very large dimensions, such as image recognition. Fine-grain massive parallelism and distributed memory, in an array of 3-transistor CID/DRAM cells, provides a throughput of $2 \times 10^{12}$ binary MACS (multiply accumulates per second) per Watt of power in a 0.5 $\mu$m process. A simple stochastic encoding scheme relaxes precision requirements in the analog implementation by one bit for each four-fold increase in vector dimension, while retaining full digital overall system resolution.

## Acknowledgments

This research was supported by ONR N00014-99-1-0612, ONR/DARPA N00014-00-C-0315, and NSF MIP-9702346. Chips were fabricated through the MOSIS service.

## Footnotes

[1]Radial basis kernels with $L_2$-norm can also be formulated in inner product format.

[2]Or, with stochastic input encoding, overflow detection could initiate a different random draw.

[3]This observation, and the binomial distribution for sums of random bits (6), forms the basis for the associative recall in a *Kanerva* memory.

## References

[1] A. Kramer, "Array-based analog computation," *IEEE Micro,* vol. **16** (5), pp. 40-49, 1996.

[2] G. Han, E. Sanchez-Sinencio, "A general purpose neuro-image processor architecture," Proc. of IEEE Int. Symp. on Circuits and Systems (ISCAS'96), vol. **3**, pp 495-498, 1996

[3] F. Kub, K. Moon, I. Mack, F. Long, "Programmable analog vector-matrix multipliers," *IEEE Journal of Solid-State Circuits,* vol. **25** (1), pp. 207-214, 1990.

[4] G. Cauwenberghs and V. Pedroni, "A Charge-Based CMOS Parallel Analog Vector Quantizer," *Adv. Neural Information Processing Systems (NIPS*94), Cambridge,* MA: MIT Press, vol. 7, pp. 779-786, 1995.

[5] Papageorgiou, C.P, Oren, M. and Poggio, T., "A General Framework for Object Detection," in *Proceedings of International Conference on Computer Vision*, 1998.

[6] G. Cauwenberghs and M.A. Bayoumi, Eds., *Learning on Silicon: Adaptive VLSI Neural Systems*, Norwell MA: Kluwer Academic, 1999.

[7] A. Murray and P.J. Edwards, "Synaptic Noise During MLP Training Enhances Fault-Tolerance, Generalization and Learning Trajectory," in *Advances in Neural Information Processing Systems,* San Mateo, CA: Morgan Kaufman, vol. **5**, pp 491-498, 1993.

[8] A. Gersho and R.M. Gray, *Vector Quantization and Signal Compression,* Norwell, MA: Kluwer, 1992.

[9] V. Vapnik, *The Nature of Statistical Learning Theory,* 2nd ed., Springer-Verlag, 1999.

[10] J. Wawrzynek, et al., "SPERT-II: A Vector Microprocessor System and its Application to Large Problems in Backpropagation Training," in *Advances in Neural Information Processing Systems,* Cambridge, MA: MIT Press, vol. 8, pp 619-625, 1996.

[11] A. Chiang, "A programmable CCD signal processor," *IEEE Journal of Solid-State Circuits,* vol. **25** (6), pp. 1510-1517, 1990.

[12] C. Neugebauer and A. Yariv, "A Parallel Analog CCD/CMOS Neural Network IC,"Proc. IEEE Int. Joint Conference on Neural Networks (IJCNN'91), Seattle, WA, vol. **1**, pp 447-451, 1991.

[13] V. Pedroni, A. Agranat, C. Neugebauer, A. Yariv, "Pattern matching and parallel processing with CCD technology," Proc. IEEE Int. Joint Conference on Neural Networks (IJCNN'92), vol. **3**, pp 620-623, 1992.

[14] M. Howes, D. Morgan, Eds., *Charge-Coupled Devices and Systems,* John Wiley & Sons, 1979.

[15] R. Genov, G. Cauwenberghs "Charge-Mode Parallel Architecture for Matrix-Vector Multiplication," *IEEE T. Circuits and Systems II,* vol. **48** (10), 2001.
